# Minimax Differential Dynamic Programming: An Application to Robust Biped Walking

**Jun Morimoto**
Human Information Science Labs,
Department 3, ATR International
Keihanna Science City,
Kyoto, JAPAN, 619-0288
xmorimo@atr.co.jp

**Christopher G. Atkeson** *
The Robotics Institute and HCII,
Carnegie Mellon University
5000 Forbes Ave.,
Pittsburgh, USA, 15213
cga@cs.cmu.edu

## Abstract

We developed a robust control policy design method in high-dimensional state space by using differential dynamic programming with a minimax criterion. As an example, we applied our method to a simulated five link biped robot. The results show lower joint torques from the optimal control policy compared to a hand-tuned PD servo controller. Results also show that the simulated biped robot can successfully walk with unknown disturbances that cause controllers generated by standard differential dynamic programming and the hand-tuned PD servo to fail. Learning to compensate for modeling error and previously unknown disturbances in conjunction with robust control design is also demonstrated.

## 1 Introduction

Reinforcement learning[8] is widely studied because of its promise to automatically generate controllers for difficult tasks from attempts to do the task. However, reinforcement learning requires a great deal of training data and computational resources, and sometimes fails to learn high dimensional tasks. To improve reinforcement learning, we propose using differential dynamic programming (DDP) which is a second order local trajectory optimization method to generate locally optimal plans and local models of the value function[2, 4]. Dynamic programming requires task models to learn tasks. However, when we apply dynamic programming to a real environment, handling inevitable modeling errors is crucial. In this study, we develop minimax differential dynamic programming which provides robust nonlinear controller designs based on the idea of $H_\infty$ control[9, 5] or risk sensitive control[6, 1]. We apply the proposed method to a simulated five link biped robot (Fig. 1). Our strategy is to use minimax DDP to find both a low torque biped walk and a policy or control law to handle deviations from the optimized trajectory. We show that both standard DDP and minimax DDP can find a local policy for lower torque biped walk than a hand-tuned PD servo controller. We show that minimax DDP can cope with larger modeling error than standard DDP or the hand-tuned PD controller. Thus, the robust controller allows us to collect useful training data. In addition, we can use learning to correct modeling

errors and model previously unknown disturbances, and design a new more optimal robust controller using additional iterations of minimax DDP.

## 2 Minimax DDP

### 2.1 Differential dynamic programming (DDP)

A value function is defined as sum of accumulated future penalty $r(\mathbf{x}_i, \mathbf{u}_i, i)$ from current state and terminal penalty $\Phi(\mathbf{x}_N)$,

$$V(\mathbf{x}_i, i) = \Phi(\mathbf{x}_N) + \sum_{j=i}^{N-1} r(\mathbf{x}_j, \mathbf{u}_j, j), \tag{1}$$

where $\mathbf{x}_i$ is the input state, $\mathbf{u}_i$ is the control output at the $i$-th time step, and $N$ is the number of time steps. Differential dynamic programming maintains a second order local model of a $Q$ function $(Q(i), Q_{\mathbf{x}}(i), Q_{\mathbf{u}}(i), Q_{\mathbf{xx}}(i), Q_{\mathbf{xu}}(i), Q_{\mathbf{uu}}(i))$, where $Q(i) = r(\mathbf{x}_i, \mathbf{u}_i, i) + V(\mathbf{x}_{i+1}, i+1)$, and the subscripts indicate partial derivatives. Then, we can derive the new control output $\mathbf{u}_i^{new} = \mathbf{u}_i + \delta \mathbf{u}_i$ from $\arg\max_{\delta \mathbf{u}_i} Q(\mathbf{x}_i + \delta \mathbf{x}_i, \mathbf{u}_i + \delta \mathbf{u}_i, i)$. Finally, by using the new control output $\mathbf{u}_i^{new}$, a second order local model of the value function $(V(i), V_{\mathbf{x}}(i), V_{\mathbf{xx}}(i))$ can be derived [2, 4].

### 2.2 Finding a local policy

DDP finds a locally optimal trajectory $\mathbf{x}_i^{opt}$ and the corresponding control trajectory $\mathbf{u}_i^{opt}$. When we apply our control algorithm to a real environment, we usually need a feedback controller to cope with unknown disturbances or modeling errors. Fortunately, DDP provides us a local policy along the optimized trajectory:

$$\mathbf{u}^{opt}(\mathbf{x}_i, i) = \mathbf{u}_i^{opt} + \mathbf{K}_i(\mathbf{x}_i - \mathbf{x}_i^{opt}), \tag{2}$$

where $\mathbf{K}_i$ is a time dependent gain matrix given by taking the derivative of the optimal policy with respect to the state [2, 4].

### 2.3 Minimax DDP

Minimax DDP can be derived as an extension of standard DDP [2, 4]. The difference is that the proposed method has an additional disturbance variable $\mathbf{w}$ to explicitly represent the existence of disturbances. This representation of the disturbance provides the robustness for optimized trajectories and policies [5].

Then, we expand the $Q$ function $Q(\mathbf{x}_i + \delta \mathbf{x}_i, \mathbf{u}_i + \delta \mathbf{u}_i, \mathbf{w}_i + \delta \mathbf{w}_i, i)$ to second order in terms of $\delta \mathbf{u}$, $\delta \mathbf{w}$ and $\delta \mathbf{x}$ about the nominal solution:

$$Q(\mathbf{x}_i + \delta \mathbf{x}_i, \mathbf{u}_i + \delta \mathbf{u}_i, \mathbf{w}_i + \delta \mathbf{w}_i, i) = Q(i) + Q_{\mathbf{x}}(i)\delta \mathbf{x}_i + Q_{\mathbf{u}}(i)\delta \mathbf{u}_i + Q_{\mathbf{w}}(i)\delta \mathbf{w}_i$$
$$+ \frac{1}{2}[\delta \mathbf{x}_i^T \delta \mathbf{u}_i^T \delta \mathbf{w}_i^T] \begin{bmatrix} Q_{\mathbf{xx}}(i) & Q_{\mathbf{xu}}(i) & Q_{\mathbf{xw}}(i) \\ Q_{\mathbf{ux}}(i) & Q_{\mathbf{uu}}(i) & Q_{\mathbf{uw}}(i) \\ Q_{\mathbf{wx}}(i) & Q_{\mathbf{wu}}(i) & Q_{\mathbf{ww}}(i) \end{bmatrix} \begin{bmatrix} \delta \mathbf{x}_i \\ \delta \mathbf{u}_i \\ \delta \mathbf{w}_i \end{bmatrix}, \tag{3}$$

The second order local model of the $Q$ function can be propagated backward in time using:

$$Q_{\mathbf{x}}(i) = V_{\mathbf{x}}(i+1)\mathbf{F}_{\mathbf{x}} + r_{\mathbf{x}}(i) \tag{4}$$

$$Q_{\mathbf{u}}(i) = V_{\mathbf{x}}(i+1)\mathbf{F}_{\mathbf{u}} + r_{\mathbf{u}}(i) \tag{5}$$

$$Q_{\mathbf{w}}(i) = V_{\mathbf{x}}(i+1)\mathbf{F}_{\mathbf{w}} + r_{\mathbf{w}}(i) \tag{6}$$

$$Q_{\mathbf{xx}}(i) = \mathbf{F}_{\mathbf{x}} V_{\mathbf{xx}}(i+1)\mathbf{F}_{\mathbf{x}} + V_{\mathbf{x}}(i+1)\mathbf{F}_{\mathbf{xx}} + r_{\mathbf{xx}}(i) \tag{7}$$

$$Q_{\mathbf{xu}}(i) = \mathbf{F_x} V_{\mathbf{xx}}(i+1)\mathbf{F_u} + V_{\mathbf{x}}(i+1)\mathbf{F_{xu}} + r_{\mathbf{xu}}(i) \qquad (8)$$
$$Q_{\mathbf{xw}}(i) = \mathbf{F_x} V_{\mathbf{xx}}(i+1)\mathbf{F_u} + V_{\mathbf{x}}(i+1)\mathbf{F_{xw}} + r_{\mathbf{xw}}(i) \qquad (9)$$
$$Q_{\mathbf{uu}}(i) = \mathbf{F_u} V_{\mathbf{xx}}(i+1)\mathbf{F_u} + V_{\mathbf{x}}(i+1)\mathbf{F_{uu}} + r_{\mathbf{uu}}(i) \qquad (10)$$
$$Q_{\mathbf{ww}}(i) = \mathbf{F_w} V_{\mathbf{xx}}(i+1)\mathbf{F_w} + V_{\mathbf{x}}(i+1)\mathbf{F_{ww}} + r_{\mathbf{ww}}(i) \qquad (11)$$
$$Q_{\mathbf{uw}}(i) = \mathbf{F_u} V_{\mathbf{xx}}(i+1)\mathbf{F_w} + V_{\mathbf{x}}(i+1)\mathbf{F_{uw}} + r_{\mathbf{uw}}(i), \qquad (12)$$

where $\mathbf{x}_{i+1} = \mathbf{F}(\mathbf{x}_i, \mathbf{u}_i, \mathbf{w}_i)$ is a model of the task dynamics.

Here, $\delta\mathbf{u}_i$ and $\delta\mathbf{w}_i$ must be chosen to minimize and maximize the second order expansion of the $Q$ function $Q(\mathbf{x}_i + \delta\mathbf{x}_i, \mathbf{u}_i + \delta\mathbf{u}_i, \mathbf{w}_i + \delta\mathbf{w}_i, i)$ in (3) respectively, i.e.,

$$\delta\mathbf{u}_i = -Q_{\mathbf{uu}}^{-1}(i)[Q_{\mathbf{ux}}(i)\delta\mathbf{x}_i + Q_{\mathbf{uw}}(i)\delta\mathbf{w}_i + Q_{\mathbf{u}}(i)]$$
$$\delta\mathbf{w}_i = -Q_{\mathbf{ww}}^{-1}(i)[Q_{\mathbf{wx}}(i)\delta\mathbf{x}_i + Q_{\mathbf{wu}}(i)\delta\mathbf{u}_i + Q_{\mathbf{w}}(i)]. \qquad (13)$$

By solving (13), we can derive both $\delta\mathbf{u}_i$ and $\delta\mathbf{w}_i$. After updating the control output $\mathbf{u}_i$ and the disturbance $\mathbf{w}_i$ with derived $\delta\mathbf{u}_i$ and $\delta\mathbf{w}_i$, the second order local model of the value function is given as

$$V(i) = V(i+1) - Q_{\mathbf{u}}(i)Q_{\mathbf{uu}}^{-1}(i)Q_{\mathbf{u}}(i) - Q_{\mathbf{w}}(i)Q_{\mathbf{ww}}^{-1}(i)Q_{\mathbf{w}}(i)$$
$$V_{\mathbf{x}}(i) = Q_{\mathbf{x}}(i) - Q_{\mathbf{u}}(i)Q_{\mathbf{uu}}^{-1}(i)Q_{\mathbf{ux}}(i) - Q_{\mathbf{w}}(i)Q_{\mathbf{ww}}^{-1}(i)Q_{\mathbf{wx}}(i)$$
$$V_{\mathbf{xx}}(i) = Q_{\mathbf{xx}}(i) - Q_{\mathbf{xu}}(i)Q_{\mathbf{uu}}^{-1}(i)Q_{\mathbf{ux}}(i) - Q_{\mathbf{xw}}(i)Q_{\mathbf{ww}}^{-1}(i)Q_{\mathbf{wx}}(i). \qquad (14)$$

## 3 Experiment

### 3.1 Biped robot model

In this paper, we use a simulated five link biped robot (Fig. 1:Left) to explore our approach. Kinematic and dynamic parameters of the simulated robot are chosen to match those of a biped robot we are currently developing (Fig. 1:Right) and which we will use to further explore our approach. Height and total weight of the robot are about 0.4 [m] and 2.0 [kg] respectively. Table 1 shows the parameters of the robot model.

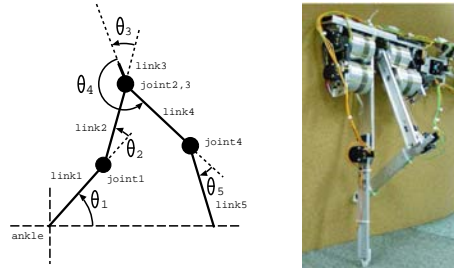

Figure 1: Left: Five link robot model, Right: Real robot

Table 1: Physical parameters of the robot model

|  | link1 | link2 | link3 | link4 | link5 |
|---|---|---|---|---|---|
| mass [kg] | 0.05 | 0.43 | 1.0 | 0.43 | 0.05 |
| length [m] | 0.2 | 0.2 | 0.01 | 0.2 | 0.2 |
| inertia [kg·m $\times 10^{-4}$] | 1.75 | 4.29 | 4.33 | 4.29 | 1.75 |

We can represent the forward dynamics of the biped robot as

$$\mathbf{x}_{i+1} = \mathbf{f}(\mathbf{x}_i) + \mathbf{b}(\mathbf{x}_i)\mathbf{u}_i, \tag{15}$$

where $\mathbf{x} = \{\theta_1, \ldots, \theta_5, \dot{\theta}_1, \ldots, \dot{\theta}_5\}$ denotes the input state vector, $\mathbf{u} = \{\tau_1, \ldots, \tau_4\}$ denotes the control command (each torque $\tau_j$ is applied to joint $j$ (Fig.1):Left). In the minimax optimization case, we explicitly represent the existence of the disturbance as

$$\mathbf{x}_{i+1} = \mathbf{f}(\mathbf{x}_i) + \mathbf{b}(\mathbf{x}_i)\mathbf{u}_i + \mathbf{b}_w(\mathbf{x}_i)\mathbf{w}_i, \tag{16}$$

where $\mathbf{w} = \{w_0, w_1, w_2, w_3, w_4\}$ denotes the disturbance ($w_0$ is applied to ankle, and $w_j$ ($j = 1 \ldots 4$) is applied to joint $j$ (Fig. 1:Left)).

## 3.2 Optimization criterion and method

We use the following objective function, which is designed to reward energy efficiency and enforce periodicity of the trajectory:

$$J = \Phi(\mathbf{x}_0, \mathbf{x}_N) + \sum_{i=0}^{N-1} r(\mathbf{x}_i, \mathbf{u}_i, i) \tag{17}$$

which is applied for half the walking cycle, from one heel strike to the next heel strike. This criterion sums the squared deviations from a nominal trajectory, the squared control magnitudes, and the squared deviations from a desired velocity of the center of mass:

$$r(\mathbf{x}_i, \mathbf{u}_i, i) = (\mathbf{x}_i - \mathbf{x}_i^d)^T Q(\mathbf{x}_i - \mathbf{x}_i^d) + \mathbf{u}_i^T R \mathbf{u}_i + (v(\mathbf{x}_i) - v^d)^T S(v(\mathbf{x}_i) - v^d), \tag{18}$$

where $\mathbf{x}_i$ is a state vector at the $i$-th time step, $\mathbf{x}_i^d$ is the nominal state vector at the $i$-th time step (taken from a trajectory generated by a hand-designed walking controller), $v(\mathbf{x}_i)$ denotes the velocity of the center of mass at the $i$-th time step, and $v^d$ denotes the desired velocity of the center of mass. The term $(\mathbf{x}_i - \mathbf{x}_i^d)^T Q(\mathbf{x}_i - \mathbf{x}_i^d)$ encourages the robot to follow the nominal trajectory, the term $\mathbf{u}_i^T R \mathbf{u}_i$ discourages using large control outputs, and the term $(v(\mathbf{x}_i) - v^d)^T S(v(\mathbf{x}_i) - v^d)$ encourages the robot to achieve the desired velocity.

In addition, penalties on the initial ($\mathbf{x}_0$) and final ($\mathbf{x}_N$) states are applied:

$$\Phi(\mathbf{x}_0, \mathbf{x}_N) = F(\mathbf{x}_0) + \Phi_N(\mathbf{x}_0, \mathbf{x}_N). \tag{19}$$

The term $F(\mathbf{x}_0)$ penalizes an initial state where the foot is not on the ground:

$$F(\mathbf{x}_0) = F_h{}^T(\mathbf{x}_0) P_0 F_h(\mathbf{x}_0), \tag{20}$$

where $F_h(\mathbf{x}_0)$ denotes height of the swing foot at the initial state $\mathbf{x}_0$. The term $\Phi_N(\mathbf{x}_0, \mathbf{x}_N)$ is used to generate periodic trajectories:

$$\Phi_N(\mathbf{x}_0, \mathbf{x}_N) = (\mathbf{x}_N - H(\mathbf{x}_0))^T P_N(\mathbf{x}_N - H(\mathbf{x}_0)), \tag{21}$$

where $\mathbf{x}_N$ denotes the terminal state, $\mathbf{x}_0$ denotes the initial state, and the term $(\mathbf{x}_N - H(\mathbf{x}_0))^T P_N (\mathbf{x}_N - H(\mathbf{x}_0))$ is a measure of terminal control accuracy. A function $H()$ represents the coordinate change caused by the exchange of a support leg and a swing leg, and the velocity change caused by a swing foot touching the ground (Appendix A).

We implement the minimax DDP by adding a minimax term to the criterion. We use a modified objective function:

$$J_{minimax} = J - \sum_{i=0}^{N-1} \mathbf{w}_i{}^T G \mathbf{w}_i, \tag{22}$$

where $\mathbf{w}_i$ denotes a disturbance vector at the $i$-th time step, and the term $\mathbf{w}_i{}^T G \mathbf{w}_i$ rewards coping with large disturbances. This explicit representation of the disturbance $\mathbf{w}$ provides the robustness for the controller [5].

## 4 Results

We compare the optimized controller with a hand-tuned PD servo controller, which also is the source of the initial and nominal trajectories in the optimization process. We set the parameters for the optimization process as $Q = 0.25\mathbf{I}_{10}$, $R = 3.0\mathbf{I}_4$, $S = 0.3\mathbf{I}_1$, desired velocity $v^d = 0.4$[m/s] in equation (18), $P_0 = 1000000.0\mathbf{I}_1$ in equation (20), and $P_N = diag\{10000.0, 10000.0, 10000.0, 10000.0, 10000.0, 10.0, 10.0, 10.0, 5.0, 5.0\}$ in equation (21), where $\mathbf{I}_N$ denotes $N$ dimensional identity matrix. For minimax DDP, we set the parameter for the disturbance reward in equation (22) as $G = diag\{5.0, 20.0, 20.0, 20.0, 20.0\}$ ($G$ with smaller elements generates more conservative but robust trajectories). Each parameter is set to acquire the best results in terms of both the robustness and the energy efficiency. When we apply the controllers acquired by standard DDP and minimax DDP to the biped walk, we adopt a local policy which we introduced in section 2.2.

Results in table 2 show that the controller generated by standard DDP and minimax DDP did almost halve the cost of the trajectory, as compared to that of the original hand-tuned PD servo controller. However, because the minimax DDP is more conservative in taking advantage of the plant dynamics, it has a slightly higher control cost than the standard DDP. Note that we defined the control cost as $\frac{1}{N}\sum_{i=0}^{N-1}||\mathbf{u}_i||^2$, where $\mathbf{u}_i$ is the control output (torque) vector at $i$-th time step, and $N$ denotes total time step for one step trajectories.

Table 2: One step control cost (average over 100 steps)

|  | PD servo | standard DDP | minimax DDP |
|---|---|---|---|
| control cost [$(N \cdot m)^2 \times 10^{-2}$ ] | 7.50 | 3.54 | 3.86 |

To test robustness, we assume that there is unknown viscous friction at each joint:

$$\tau_j^{dist} = -\mu_j \dot{\theta}_j \quad (j = 1, \ldots, 4), \tag{23}$$

where $\mu_j$ denotes the viscous friction coefficient at joint $j$.

We used two levels of disturbances in the simulation, with the higher level being 3 times larger than the base level (Table 3).

Table 3: Parameters of the disturbance

|  | $\mu_2, \mu_3$ (hip joints) | $\mu_1, \mu_4$ (knee joints) |
|---|---|---|
| base | 0.01 | 0.05 |
| large | 0.03 | 0.15 |

All methods could handle the base level disturbances. Both the standard and the minimax DDP generated much less control cost than the hand-tuned PD servo controller (Table 4). However, only the minimax DDP control design could cope with the higher level of disturbances. Figure 2 shows trajectories for the three different methods. Both the simulated robot with the standard DDP and the hand-tuned PD servo controller fell down before achieving 100 steps. The bottom of figure 2 shows part of a successful biped walking trajectory of the robot with the minimax DDP. Figure 3 shows ankle joint trajectories for the three different methods. Only the minimax DDP successfully kept ankle joint $\theta_1$ around 90 degrees more than 20 seconds. Table 5 shows the number of steps before the robot fell down. We terminated a trial when the robot achieved 1000 steps.

Table 4: One step control cost with the base setting (averaged over 100 steps)

| | PD servo | standard DDP | minimax DDP |
|---|---|---|---|
| control cost $[(N \cdot m)^2 \times 10^{-2}]$ | 8.97 | 5.23 | 5.87 |

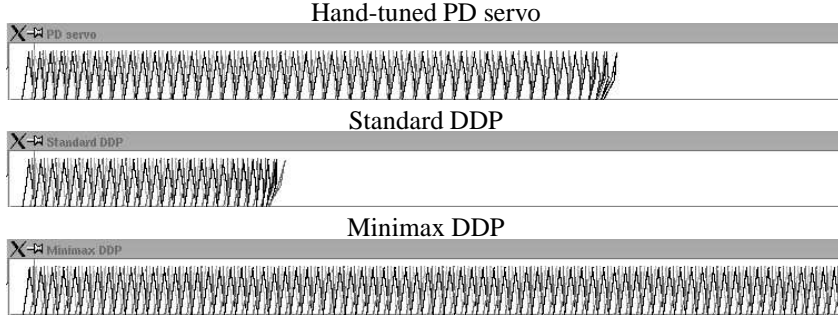

Figure 2: Biped walk trajectories with the three different methods

## 5 Learning the unmodeled dynamics

In section 4, we verified that minimax DDP could generate robust biped trajectories and policies. The minimax DDP coped with larger disturbances than the standard DDP and the hand-tuned PD servo controller. However, if there are modeling errors, using a robust controller which does not learn is not particularly energy efficient. Fortunately, with minimax DDP, we can collect sufficient data to improve our dynamics model. Here, we propose using Receptive Field Weighted Regression (RFWR) [7] to learn the error dynamics of the biped robot. In this section we present results on learning a *simulated* modeling error (the disturbances discussed in section 4). We are currently applying this approach to an actual robot.

We can represent the full dynamics as the sum of the known dynamics and the error dynamics $\Delta\mathbf{F}(\mathbf{x}_i, \mathbf{u}_i, i)$:

$$\mathbf{x}_{i+1} = \mathbf{F}(\mathbf{x}_i, \mathbf{u}_i) + \Delta\mathbf{F}(\mathbf{x}_i, \mathbf{u}_i, i). \tag{24}$$

We estimate the error dynamics $\Delta\mathbf{F}$ by using RFWR:

$$\Delta\hat{\mathbf{F}}(\mathbf{x}_i, \mathbf{u}_i, i) = \frac{\sum_{k=1}^{N_b} \alpha_k^i \phi_k(\mathbf{x}_i, \mathbf{u}_i, i)}{\sum_{k=1}^{N_b} \alpha_k^i}, \tag{25}$$

$$\phi_k(\mathbf{x}_i, \mathbf{u}_i, i) = \beta_k^T \tilde{\mathbf{x}}_k^i, \tag{26}$$

$$\alpha_k^i = \exp\left(-\frac{1}{2}(i - c_k) D_k (i - c_k)\right), \tag{27}$$

where, $N_b$ denotes the number of basis function, $c_k$ denotes center of $k$-th basis function, $D_k$ denotes distance metric of the $k$-th basis function, $\beta_k$ denotes parameter of the $k$-th basis function to approximate error dynamics, and $\tilde{\mathbf{x}}_k^i = (\mathbf{x}_i, \mathbf{u}_i, 1, i - c_k)$ denotes augmented state vector for the $k$-th basis function. We align 20 basis functions ($N_b = 20$) at even intervals along the biped trajectories.

The learning strategy uses the following sequence: 1) Design the initial controller using minimax DDP applied to the nominal model. 2) Apply that controller. 3) Learn the actual dynamics using RFWR. 4) Redesign the biped controller using minimax DDP with the learned model.

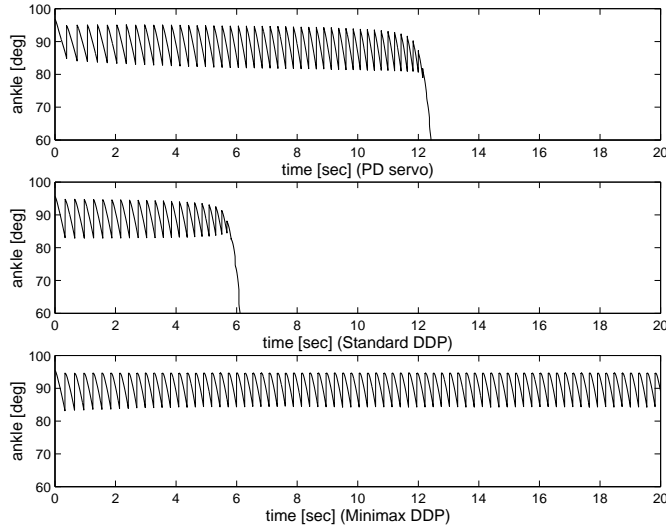

Figure 3: Ankle joint trajectories with the three different methods

Table 5: Number of steps with the large disturbances

|  | PD servo | standard DDP | minimax DDP |
|---|---|---|---|
| Number of steps | 49 | 24 | $> 1000$ |

We compare the efficiency of the controller with the learned model to the controller without the learned model. Results in table 6 show that the controller after learning the error dynamics used lower torque to produce stable biped walking trajectories.

Table 6: One step control cost with the large disturbances (averaged over 100 steps)

|  | without learned model | with learned model |
|---|---|---|
| control cost $[(N \cdot m)^2 \times 10^{-2}]$ | 17.1 | 11.3 |

## 6   Discussion

In this study, we developed an optimization method to generate biped walking trajectories by using differential dynamic programming (DDP). We showed that 1) DDP and minimax DDP can be applied to high dimensional problems, 2) minimax DDP can design more robust controllers, and 3) learning can be used to reduce modeling error and unknown disturbances in the context of minimax DDP control design.

Both standard DDP and minimax DDP generated low torque biped trajectories. We showed that the minimax DDP control design was more robust than the controller designed by standard DDP and the hand-tuned PD servo. Given a robust controller, we could collect sufficient data to learn the error dynamics using RFWR[7] without the robot falling down all the time. We also showed that after learning the error dynamics, the biped robot could find a lower torque trajectory.

DDP provides a feedback controller which is important in coping with unknown distur-

bances and modeling errors. However, as shown in equation (2), the feedback controller is indexed by time, and development of a time independent feedback controller is a future goal.

**Appendix**

## A   Ground contact model

The function $H()$ in equation (21) includes the mapping (velocity change) caused by ground contact. To derive the first derivative of the value function $V_{\mathbf{x}}(\mathbf{x}_N)$ and the second derivative $V_{\mathbf{xx}}(\mathbf{x}_N)$, where $\mathbf{x}_N$ denotes the terminal state, the function $H()$ should be analytical. Then, we used an analytical ground contact model[3]:

$$\dot{\boldsymbol{\theta}}^+ - \dot{\boldsymbol{\theta}}^- = M^{-1}(\boldsymbol{\theta})D(\boldsymbol{\theta})\mathbf{f}\Delta t, \tag{28}$$

where $\boldsymbol{\theta}$ denotes joint angles of the robot, $\dot{\boldsymbol{\theta}}^-$ denotes angular velocities before ground contact, $\dot{\boldsymbol{\theta}}^+$ denotes angular velocities after ground contact, $M$ denotes the inertia matrix, D denotes the Jacobian matrix which converts the ground contact force $\mathbf{f}$ to the torque at each joint, and $\Delta t$ denotes time step of the simulation.

## Footnotes

*also affiliated with Human Information Science Laboratories, Department 3, ATR International

## References

[1] S. P. Coraluppi and S. I. Marcus. Risk-Sensitive and Minmax Control of Discrete-Time Finite-State Markov Decision Processes. *Automatica*, 35:301–309, 1999.

[2] P. Dyer and S. R. McReynolds. *The Computation and Theory of Optimal Control*. Academic Press, New York, NY, 1970.

[3] Y. Hurmuzlu and D. B. Marghitu. Rigid body collisions of planar kinematic chains with multiple contact points. *International Journal of Robotics Research*, 13(1):82–92, 1994.

[4] D. H. Jacobson and D. Q. Mayne. *Differential Dynamic Programming*. Elsevier, New York, NY, 1970.

[5] J. Morimoto and K. Doya. Robust Reinforcement Learning. In Todd K. Leen, Thomas G. Dietterich, and Volker Tresp, editors, *Advances in Neural Information Processing Systems 13*, pages 1061–1067. MIT Press, Cambridge, MA, 2001.

[6] R. Neuneier and O. Mihatsch. Risk Sensitive Reinforcement Learning. In M. S. Kearns, S. A. Solla, and D. A. Cohn, editors, *Advances in Neural Information Processing Systems 11*, pages 1031–1037. MIT Press, Cambridge, MA, USA, 1998.

[7] S. Schaal and C. G. Atkeson. Constructive incremental learning from only local information. *Neural Computation*, 10(8):2047–2084, 1998.

[8] R. S. Sutton and A. G. Barto. *Reinforcement Learning: An Introduction*. The MIT Press, Cambridge, MA, 1998.

[9] K. Zhou, J. C. Doyle, and K. Glover. *Robust Optimal Control*. PRENTICE HALL, New Jersey, 1996.
